# Probabilistic Image Sensor Fusion

Ravi K. Sharma[1], Todd K. Leen[2] and Misha Pavel[1]

[1]Department of Electrical and Computer Engineering
[2]Department of Computer Science and Engineering
Oregon Graduate Institute of Science and Technology
P.O. Box 91000, Portland, OR 97291-1000
Email: {ravi,pavel}@ece.ogi.edu, tleen@cse.ogi.edu

## Abstract

We present a probabilistic method for fusion of images produced by multiple sensors. The approach is based on an image formation model in which the sensor images are noisy, locally linear functions of an *underlying, true scene*. A Bayesian framework then provides for maximum likelihood or maximum a posteriori estimates of the true scene from the sensor images. Maximum likelihood estimates of the parameters of the image formation model involve (local) second order image statistics, and thus are related to local principal component analysis. We demonstrate the efficacy of the method on images from visible-band and infrared sensors.

## 1 Introduction

Advances in sensing devices have fueled the deployment of multiple sensors in several computational vision systems [1, for example]. Using multiple sensors can increase reliability with respect to single sensor systems. This work was motivated by a need for an aircraft autonomous landing guidance (ALG) system [2, 3] that uses visible-band, infrared (IR) and radar-based imaging sensors to provide guidance to pilots for landing aircraft in low visibility. IR is suitable for night operation, whereas radar can penetrate fog. The application requires fusion algorithms [4] to combine the different sensor images.

Images from different sensors have different characteristics arising from the varied physical imaging processes. Local contrast may be polarity reversed between visible-band and IR images [5, 6]. A particular sensor image may contain local features not found in another sensor image, i.e., sensors may report complementary features. Finally, individual sensors are subject to noise. Fig. 1(a) and 1(b) are visible-band and IR images respectively, of a runway scene showing polarity reversed (rectangle)

and complementary (circle) features. These effects pose difficulties for fusion.

An obvious approach to fusion is to average the pixel intensities from different sensors. Averaging, Fig. 1(c), increases the signal to noise ratio, but reduces the contrast where there are polarity reversed or complementary features [7].

Transform-based fusion methods [8, 5, 9] *select* from one sensor or another for fusion. They consist of three steps: (i) decompose the sensor images using a specified transform e.g. a multiresolution Laplacian pyramid, (ii) fuse at each level of the pyramid by selecting the highest energy transform coefficient, and (iii) invert the transform to synthesize the fused image. Since features are selected rather than averaged, they are rendered at full contrast, but the methods are sensitive to sensor noise, see Fig. 1(d).

To overcome the limitations of averaging or selection methods, and put sensor fusion on firm theoretical grounds, we explicitly model the production of sensor images from the true scene, including the effects of sensor noise. From the model, and sensor images, one can ask *What is the most probable true scene?* This forms the basis for fusing the sensor images. Our technique uses the Laplacian pyramid representation [5], with the step (ii) above replaced by our probabilistic fusion. A similar probabilistic framework for sensor fusion is discussed in ([10]).

## 2   The Image Formation Model

The true scene, denoted $s$, gives rise to a sensor image through a noisy, non-linear transformation. For ALG, $s$ would be an image of the landing scene under conditions of uniform lighting, unlimited visibility, and perfect sensors. We model the map from the true scene to a sensor image by a noisy, *locally affine transformation* whose parameters are allowed to vary across the image (actually across the Laplacian pyramid)

$$a_i(\vec{l},t) = \beta_i(\vec{l},t)\ s(\vec{l},t) + \alpha_i(\vec{l},t) + \epsilon_i(\vec{l},t) \tag{1}$$

where, $s$ is the true scene, $a_i$ is $i^{th}$ sensor image, $\vec{l} \equiv (x,y,k)$ is the hyperpixel location, with $x,y$ the pixel coordinates and $k$ the level of the pyramid, $t$ is the time, $\alpha$ is the sensor offset, $\beta$ is the sensor gain (which includes the effects of local polarity reversals and complementarity), and $\epsilon$ is the (zero-mean) sensor noise. To simplify notation, we adopt the matrix form

$$a = \beta s + \alpha + \epsilon \tag{2}$$

where $a = [a_1, a_2, \ldots, a_q]^{\mathrm{T}}$, $\beta = [\beta_1, \beta_2, \ldots, \beta_q]^{\mathrm{T}}$, $\alpha = [\alpha_1, \alpha_2, \ldots, \alpha_q]^{\mathrm{T}}$, $s$ is a scalar and $\epsilon = [\epsilon_1, \epsilon_2, ..., \epsilon_q]^{\mathrm{T}}$, and we have dropped the reference to location and time.

Since the image formation parameters $\beta$, $\alpha$, and the sensor noise covariance $\Sigma_\epsilon$ can vary from hyperpixel to hyperpixel, the model can express local polarity reversals, complementary features, spatial variation of sensor gain, and noise.

We do assume, however, that the image formation parameters and sensor noise distribution vary *slowly* with location[1]. Hence, a particular set of parameters is considered to hold true over a spatial region of several square hyperpixels. We will use this assumption implicitly when we estimate these parameters from data.

The model (2) fits the framework of the factor analysis model in statistics [11, 12]. Here the hyperpixel values of the true scene $s$ are the latent variables or

common factors, $\beta$ contains the factor loadings, and the sensor noise $\epsilon$ values are the independent factors. Estimation of the true scene is equivalent to estimating the common factors from the observations $a$.

## 3    Bayesian Fusion

Given the sensor intensities $a$, we will estimate the true scene $s$ by appeal to a Bayesian framework. We assume that the probability density function of the latent variables $s$ is a Gaussian with local mean $s_0(\vec{l}, t)$ and local variance $\sigma_s^2(\vec{l}, t)$. An attractive benefit of this setup is that the *prior mean* $s_0$ might be obtained from knowledge in the form of maps, or clear-weather images of the scene. Thus, such database information can be folded into the sensor fusion in a natural way.

The density on the sensor images conditioned on the true scene, $\mathcal{P}(a|s)$, is normal with mean $\beta s + \alpha$ and covariance $\Sigma_\epsilon = \mathrm{diag}[\sigma_{\epsilon_1}^2, \sigma_{\epsilon_2}^2, \ldots, \sigma_{\epsilon_q}^2]$. The marginal density $\mathcal{P}(a)$ is normal with mean $\boldsymbol{\mu}_m = \beta s_0 + \alpha$ and covariance

$$\mathbf{C} = \Sigma_\epsilon + \sigma_s^2 \beta \beta^{\mathrm{T}} \tag{3}$$

Finally, the posterior density on $s$, given the sensor data $a$, $\mathcal{P}(s|a)$ is also normal with mean $\mathbf{M}^{-1}(\beta^{\mathrm{T}} \Sigma_\epsilon^{-1}(a - \alpha) + s_0/\sigma_s^2)$, and covariance $\mathbf{M}^{-1} = (\beta^{\mathrm{T}} \Sigma_\epsilon^{-1} \beta + 1/\sigma_s^2)^{-1}$.

Given these densities, there are two obvious candidates for probabilistic fusion: *maximum likelihood* (ML) $\hat{s} = \max_s \mathcal{P}(a|s)$, and *maximum a posteriori* (MAP) $\hat{s} = \max_s \mathcal{P}(s|a)$.

The MAP fusion estimate is simply the posterior mean

$$\hat{s} = [\beta^{\mathrm{T}} \Sigma_\epsilon^{-1} \beta + 1/\sigma_s^2]^{-1} \left( \beta^{\mathrm{T}} \Sigma_\epsilon^{-1}(a - \alpha) + s_0/\sigma_s^2 \right) \tag{4}$$

which for two sensors reads

$$\hat{s} = \left( \frac{\beta_1(a_1 - \alpha_1)}{\sigma_{\epsilon_1}^2} + \frac{\beta_2(a_2 - \alpha_2)}{\sigma_{\epsilon_2}^2} + \frac{s_0}{\sigma_s^2} \right) \bigg/ \left( \frac{\beta_1^2}{\sigma_{\epsilon_1}^2} + \frac{\beta_2^2}{\sigma_{\epsilon_2}^2} + \frac{1}{\sigma_s^2} \right) . \tag{5}$$

To obtain the ML fusion estimate we take the limit $\sigma_s^2 \to \infty$ in either (4) or (5).

For both ML and MAP, the fused image $\hat{s}$ is a locally linear combination of the sensor images that can, through the spatio-temporal variations in $\beta$, $\alpha$, and $\Sigma_\epsilon$, properly respond to changes in the sensor characteristics that tax averaging or selection schemes. For example, if the second sensor has a polarity reversal relative to the first, then $\beta_2$ is negative and the two sensor contributions are properly *subtracted*. If the first sensor has high noise (large $\sigma_{\epsilon_1}^2$), its contribution to the fused image is attenuated. Finally, a feature missing from sensor 1 corresponds to $\beta_1 = 0$. The model compensates by accentuating the contribution from sensor 2.

## 4    Model Parameter Estimates

We need to estimate the local image formation model parameters $\alpha(\vec{l}, t), \beta(\vec{l}, t)$ and the local sensor noise covariance $\dot{\Sigma}_\epsilon(\vec{l}, t)$. We estimate the latter from successive, motion compensated video frames from each sensor. First we estimate the average value at each hyperpixel $(\overline{a_i}(t))$, and the average square $(\overline{a_i^2}(t))$ by exponential moving averages. We next estimate the noise variance by the difference $\sigma_{\epsilon_i}^2(t) = \overline{a_i^2}(t) - \overline{a_i}^2(t)$.

To estimate $\beta$ and $\alpha$, we assume that $\beta$, $\alpha$, $\Sigma_\epsilon$, $s_0$ and $\sigma_s^2$ are nearly constant over small spatial regions ($5 \times 5$ blocks) surrounding the hyperpixel for which the

parameters are desired. Essentially we are invoking a spatial analog of ergodicity, where ensemble averages are replaced by spatial averages, carried out locally over regions in which the statistics are approximately constant.

To form a maximum likelihood (ML) estimate of $\boldsymbol{\alpha}$, we extremize the data log-likelihood $\mathcal{L} = \sum_{n=1}^{N} \log[\mathcal{P}(\boldsymbol{a}_n)]$ with respect to $\boldsymbol{\alpha}$ to obtain

$$\boldsymbol{\alpha}_{\mathrm{ML}} = \boldsymbol{\mu}_a - \boldsymbol{\beta} s_0 \quad , \tag{6}$$

where $\boldsymbol{\mu}_a$ is the data mean, computed over a $5 \times 5$ hyperpixel local region ($N = 25$ points).

To obtain a ML estimate of $\boldsymbol{\beta}$, we set the derivatives of $\mathcal{L}$ with respect to $\boldsymbol{\beta}$ equal to zero and recover

$$(\mathbf{C} - \boldsymbol{\Sigma}_a)\mathbf{C}^{-1}\boldsymbol{\beta} = 0 \tag{7}$$

where $\boldsymbol{\Sigma}_a$ is the data covariance matrix, also computed over a $5 \times 5$ hyperpixel local region. The only non-trivial solution to (7) is

$$\boldsymbol{\beta}_{\mathrm{ML}} = \boldsymbol{\Sigma}_\epsilon^{\frac{1}{2}} \tilde{\mathbf{U}} \frac{(\tilde{\lambda} - 1)^{\frac{1}{2}}}{\sigma_s} r \tag{8}$$

where $\tilde{\mathbf{U}}$, $\tilde{\lambda}$ are the principal eigenvector and eigenvalue of the weighted data covariance matrix, $\tilde{\boldsymbol{\Sigma}}_a \equiv \boldsymbol{\Sigma}_\epsilon^{-\frac{1}{2}} \boldsymbol{\Sigma}_a \boldsymbol{\Sigma}_\epsilon^{-\frac{1}{2}}$, and $r = \pm 1$.

An alternative to maximum likelihood estimation is the least squares (LS) approach [11]. We obtain the LS estimate $\boldsymbol{\alpha}_{\mathrm{LS}}$ by minimizing

$$E_\alpha = \| \boldsymbol{\mu}_a - \boldsymbol{\mu}_m \|^2 \tag{9}$$

with respect to $\boldsymbol{\alpha}$. This gives

$$\boldsymbol{\alpha}_{\mathrm{LS}} = \boldsymbol{\mu}_a - \boldsymbol{\beta} s_0 \quad . \tag{10}$$

The least squares estimate $\boldsymbol{\beta}_{\mathrm{LS}}$ is obtained by minimizing

$$E_\beta = \| \boldsymbol{\Sigma}_a - \mathbf{C} \|^2 \tag{11}$$

with respect to $\boldsymbol{\beta}$. The solution to this minimization is

$$\boldsymbol{\beta}_{\mathrm{LS}} = \frac{\lambda^{\frac{1}{2}}}{\sigma_s} \mathbf{U} r \tag{12}$$

where $\mathbf{U}$, $\lambda$ are the principal eigenvector and eigenvalue of the noise-corrected covariance matrix $(\boldsymbol{\Sigma}_a - \boldsymbol{\Sigma}_\epsilon)$, and $r = \pm 1$.[2]

The estimation procedures cannot provide values of the priors $\sigma_s^2$ and $s_0$. Were we dealing with a single global model, this would pose no problem. But we must impose a constraint in order to smoothly piece together our local models. We impose that $\|\boldsymbol{\beta}\| = 1$ everywhere, or by (12) $\sigma_s^2 = \lambda$. Recall that $\lambda$ is the leading eigenvalue of $\boldsymbol{\Sigma}_a - \boldsymbol{\Sigma}_\epsilon$ and thus captures the scale of variations in $\boldsymbol{a}$ that arise from variations in $s$. Thus we would expect $\lambda \propto \sigma_s^2$. Our constraint insures that the proportionality constant be the same for each local model. Next, note that changing $s_0$ causes a shift

in $\hat{s}$. To maintain consistency between local regions, we take $s_0 = 0$ everywhere. These choices for $\sigma_s^2$ and $s_0$ constrain the parameter estimates to

$$\beta_{\text{LS}} = r \, \mathbf{U} \quad \text{and}$$
$$\alpha_{\text{LS}} = \boldsymbol{\mu}_a \, . \tag{13}$$

In (5) $\sigma_s^2$ and $s_0$ are defined at each hyperpixel. However, to estimate $\beta$ and $\boldsymbol{\alpha}$, we used spatial averages to compute the sample mean and covariance. This is somewhat inconsistent, since the spatial variation of $s_0$ (e.g. when there are edges in the scene) is not explicitly captured in the model mean and covariance. These variations are, instead, attributed to $\sigma_s^2$, resulting in overestimation of the latter. A more complete model would explicitly model the spatial variations of $s_0$, though we expect this will produce only minor changes in the results.

Finally, the sign parameter $r$ is not specified. In order to properly piece together our local models, we must choose $r$ at each hyperpixel in such a way that $\beta$ changes direction *slowly* as we move from hyperpixel to hyperpixel and encounter changes in the local image statistics. That is, large direction changes due to arbitrary sign reversals are not allowed. We use a simple heuristic to accomplish this.

## 5  Relation to PCA

The MAP and ML fusion rules are closely related to PCA. To see this, assume that the noise is homoscedastic $\boldsymbol{\Sigma}_\epsilon = \sigma_\epsilon^2 \mathbf{I}$ and use the parameter estimates (13) in the MAP fusion rule (5), reducing the latter to

$$\hat{s} = \frac{1}{1 + \sigma_\epsilon^2/\sigma_s^2} \, \mathbf{U}_a^{\mathsf{T}} (\boldsymbol{a} - \boldsymbol{\mu}_a) + \frac{1}{1 + \sigma_s^2/\sigma_\epsilon^2} \, s_0 \tag{14}$$

where $\mathbf{U}_a$ is the principal eigenvector of the data covariance matrix $\boldsymbol{\Sigma}_a$. The MAP estimate $\hat{s}$ is simply a scaled and shifted local PCA projection of the sensor data.

Both the scaling and shift arise because the prior distribution on $s$ tends to bias $\hat{s}$ towards $s_0$. When the prior is flat $\sigma_s^2 \to \infty$, (or equivalently when using the ML fusion estimate), or when the noise variance vanishes, the fused image is given by a simple local PCA projection

$$\hat{s} = \mathbf{U}_a^{\mathsf{T}} (\boldsymbol{a} - \boldsymbol{\mu}_a) \quad . \tag{15}$$

## 6  Experiments and Results

We applied our fusion method to visible-band and IR runway images, Fig. 1, containing additive Gaussian noise. Fig. 1(e) shows the result of ML fusion with $\beta$ and $\boldsymbol{\alpha}$ estimated using (13). ML fusion performs better than either averaging or selection in regions that contain local polarity reversals or complementary features. ML fusion gives higher weight to IR in regions where the features in the two images are common, thus reducing the effects of noise in the visible-band image. ML fusion gives higher weight to the appropriate sensor in regions with complementary features.

Fig. 1(f) shows the result of MAP fusion (5) with the priors $\sigma_s^2$ and $s_0$ those dictated by the consistency requirements discussed in section 4. Clearly, the MAP image is less noisy than the ML image. In regions of low sensor image contrast, $\sigma_s^2$ is low (since $\lambda$ is low), thus the contribution from the sensor images is attenuated compared to the ML fusion rule. Hence the noise is attenuated. In regions containing features such as edges, $\sigma_s^2$ is high (since $\lambda$ is high); hence the contribution from the sensor images is similar to that in ML fusion.

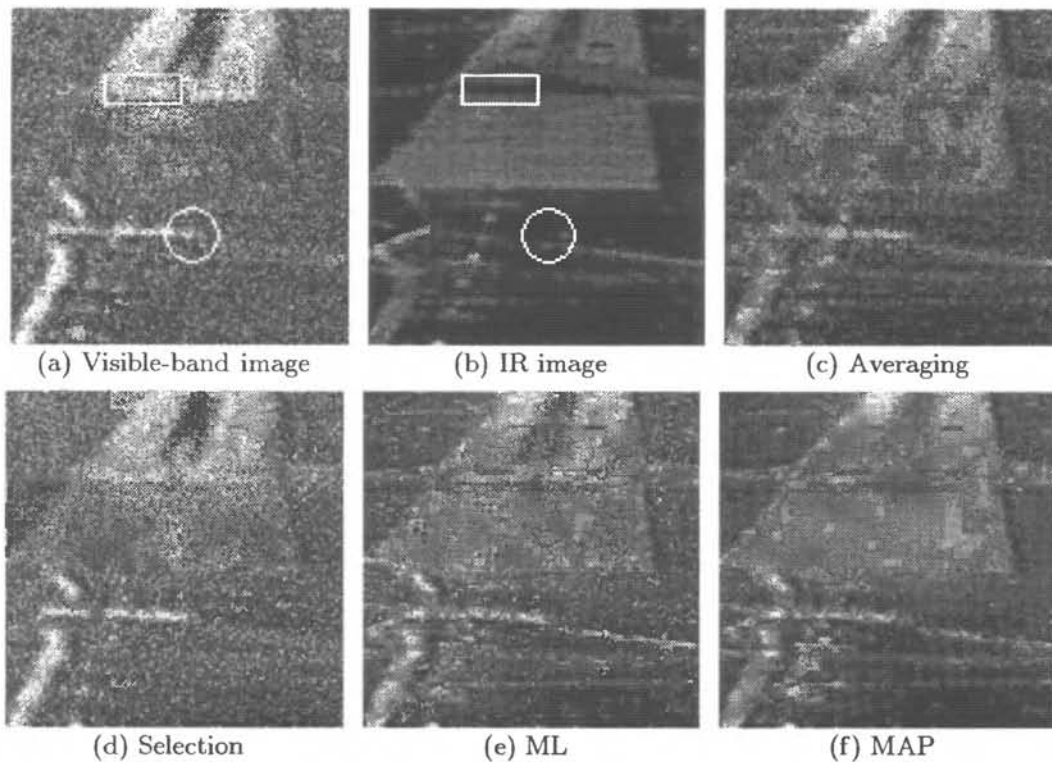

| | | |
|---|---|---|
| (a) Visible-band image | (b) IR image | (c) Averaging |
| (d) Selection | (e) ML | (f) MAP |

Figure 1: Fusion of visible-band and IR images containing additive Gaussian noise

In Fig. 2 we demonstrate the use of a database image for fusion. Fig. 2(a) and 2(b) are simulated noisy sensor images from visible-band and IR, that depict a runway with an aircraft on it. Fig. 2(c) is an image of the same scene as might be obtained from a terrain database. Although this image is clean, it does not show the actual situation on the runway. One can use the database image pixel intensities as the prior mean $s_0$ in the MAP fusion rule (5). The prior variance $\sigma_s^2$ in (5) can be regarded as a measure of confidence in the database image – it's value controls the relative contribution of the sensors vs. the database image in the fused image. (The parameters $\beta$ and $\alpha$, and the sensor noise covariance $\Sigma_\epsilon$ were estimated exactly as before.) Fig. 2(d), 2(e) and 2(f) show the MAP-fused image as a function of increasing $\sigma_s^2$. Higher values of $\sigma_s^2$ accentuate the contribution of the sensor images, whereas lower values of $\sigma_s^2$ accentuate the contribution of the database.

## 7   Discussion

We presented a model-based probabilistic framework for fusion of images from multiple sensors and exercised the approach on visible-band and IR images. The approach provides both a rigorous framework for PCA-like fusion rules, and a principled way to combine information from a terrain database with sensor images.

We envision several refinements of the approach given here. Writing new image formation models at each hyperpixel produces an overabundance of models. Early experiments show that this can be relaxed by using the same model parameters over regions of several square hyperpixels, rather than recalculating for each hyperpixel. A further refinement could be provided by adopting a mixture of linear models to build up the non-linear image formation model. Finally, we have used multiple frames from a video sequence to obtain ML and MAP fused sequences, and one should be able to produce superior parameter estimates by suitable use of the video sequence.

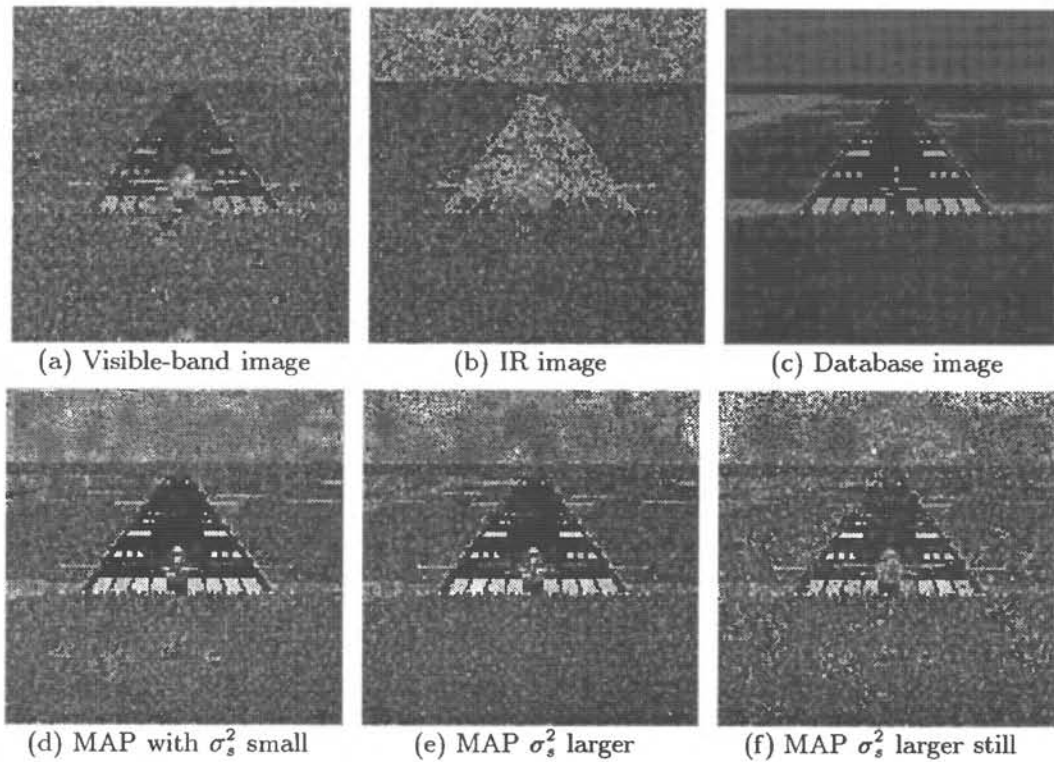

(a) Visible-band image        (b) IR image        (c) Database image

(d) MAP with $\sigma_s^2$ small        (e) MAP $\sigma_s^2$ larger        (f) MAP $\sigma_s^2$ larger still

Figure 2: Fusion of simulated visible-band and IR images using database image

**Acknowledgments** – This work was supported by NASA Ames Research Center grant NCC2-811. TKL was partially supported by NSF grant ECS-9704094.

## Footnotes

[1]Specifically the parameters vary slowly on the spatio-temporal scales over which the true scene $s$ may exhibit large variations.

[2]The least squares and maximum likelihood solutions are *identical* when the model is exact $\boldsymbol{\Sigma}_a \equiv \mathbf{C}$, i.e. the observed data covariance is *exactly* of the form dictated by the model. Under this condition, $\tilde{\mathbf{U}} = (\mathbf{U}^\mathrm{T}\boldsymbol{\Sigma}_\epsilon^{-1}\mathbf{U})^{-1/2}\boldsymbol{\Sigma}_\epsilon^{-1/2}\mathbf{U}$ and $(\tilde{\lambda} - 1) = \lambda(\mathbf{U}^\mathrm{T}\boldsymbol{\Sigma}_\epsilon^{-1}\mathbf{U})$. The LS and ML solutions are also identical when the noise covariance is homoscedastic $\boldsymbol{\Sigma}_\epsilon = \sigma_\epsilon^2 \mathbf{I}$, even if the model is *not* exact.

# References

[1] L. A. Klein. *Sensor and Data Fusion Concepts and Applications*. SPIE, 1993.

[2] J. R. Kerr, D. P. Pond, and S. Inman. Infrared-optical multisensor for autonomous landing guidance. *Proceedings of SPIE*, 2463:38–45, 1995.

[3] B. Roberts and P. Symosek. Image processing for flight crew situation awareness. *Proceedings of SPIE*, 2220:246–255, 1994.

[4] M. Pavel and R. K. Sharma. Model-based sensor fusion for aviation. In J. G. Verly, editor, *Enhanced and Synthetic Vision 1997*, volume 3088, pages 169–176. SPIE, 1997.

[5] P. J. Burt and R. J. Kolczynski. Enhanced image capture through fusion. In *Fourth Int. Conf. on Computer Vision*, pages 173–182. IEEE Comp. Soc., 1993.

[6] H. Li and Y. Zhou. Automatic visual/IR image registration. *Optical Engineering*, 35(2):391–400, 1996.

[7] M. Pavel, J. Larimer, and A. Ahumada. Sensor fusion for synthetic vision. In *Proceedings of the Society for Information Display*, pages 475–478. SPIE, 1992.

[8] P. Burt. A gradient pyramid basis for pattern-selective image fusion. In *Proceedings of the Society for Information Display*, pages 467–470. SPIE, 1992.

[9] A. Toet. Hierarchical image fusion. *Machine Vision and Applications*, 3:1–11, 1990.

[10] J. J. Clark and A. L. Yuille. *Data Fusion for Sensory Information Processing Systems*. Kluwer, Boston, 1990.

[11] A. Basilevsky. *Statistical Factor Analysis and Related Methods*. Wiley, 1994.

[12] M. E. Tipping and C. M. Bishop. Probabilistic principal component analysis. Technical report, NCRG/97/010, Neural Computing Research Group, Aston University, UK, 1997.
